# Handling Missing Data with Variational Bayesian Learning of ICA

**Kwokleung Chan, Te-Won Lee and Terrence Sejnowski**
The Salk Institute, Computational Neurobiology Laboratory,
10010 N. Torrey Pines Road,
La Jolla,, CA 92037, USA
{*kwchan,tewon,terry*}*@salk.edu*

## Abstract

Missing data is common in real-world datasets and is a problem for many estimation techniques. We have developed a variational Bayesian method to perform Independent Component Analysis (ICA) on high-dimensional data containing missing entries. Missing data are handled naturally in the Bayesian framework by integrating the generative density model. Modeling the distributions of the independent sources with mixture of Gaussians allows sources to be estimated with different kurtosis and skewness. The variational Bayesian method automatically determines the dimensionality of the data and yields an accurate density model for the observed data without overfitting problems. This allows direct probability estimation of missing values in the high dimensional space and avoids dimension reduction preprocessing which is not feasible with missing data.

## 1 Introduction

Data density estimation is an important step in many machine learning problems. Often we are faced with data containing incomplete entries. The data may be missing due to measurement or recording failure. Another frequent cause is difficulty in collecting complete data. For example, it could be expensive and time consuming to perform some biomedical tests. Data scarcity is not uncommon and it would be very undesirable to discard those data points with missing entries when we already have a small dataset. Traditionally, missing data are filled in by *mean imputation* or *regression imputation* during preprocessing. This could introduce biases into the data cloud density and adversely affect subsequent analysis. A more principled way would be to use probability density estimates of the missing entries instead of point estimates. A well known example of this approach is the use of Expectation-Maximization (EM) algorithm in fitting incomplete data with a single Gaussian density [5].

Independent Component Analysis (ICA) [4] tries to locate independent axes within the data cloud and was developed for blind source separation. It has been applied to speech separation and analyzing fMRI and EEG data. ICA is also used to model data density, describing data as linear mixture of independent features and finding projections that may uncover interesting structure in the data. Maximum likelihood learning of ICA with incomplete data has been studied by [6], in the limited case of a square mixing matrix and predefined source densities.

Many real-world datasets have intrinsic dimensionality smaller then that of the observed

data. With missing data, principal component analysis cannot be used to perform dimension reduction as preprocessing for ICA. Instead, the variational Bayesian method applied to ICA can handle small datasets with high observed dimension [1, 2]. The Bayesian method prevents overfitting and performs automatic dimension reduction. In this paper, we extend the variational Bayesian ICA method to problems with missing data. The probability density estimate of the missing entries can be used to fill in the missing values. This also allows the density model to be refined and made more accurate.

## 2 Model and Theory

### 2.1 ICA generative model with missing data

Consider a data set of $T$ data points in an $N$-dimensional space: $\mathbf{X} = \{\mathbf{x}_t \in \mathcal{R}^N\}$, $t = \{1, \cdots, T\}$. Assume a noisy ICA generative model for the data:

$$P(\mathbf{x}_t|\theta) = \int \mathcal{N}(\mathbf{x}_t|\mathbf{A}\mathbf{s}_t + \boldsymbol{\nu}, \boldsymbol{\Psi})P(\mathbf{s}_t|\theta_s)\, d\mathbf{s}_t \tag{1}$$

where $\mathbf{A}$ is the mixing matrix, $\boldsymbol{\nu}$ is the observation mean and $\boldsymbol{\Psi}^{-1}$ is the diagonal noise variance. The hidden source $\mathbf{s}_t$ is assumed to have $L$ dimensions. Each component of $\mathbf{s}_t$ is modeled by a mixture of $K$ Gaussians to allow for source densities of various kurtosis and skewness,

$$P(\mathbf{s}_t|\theta_s) \quad = \quad \prod_l^L \left( \sum_{k_l}^K \pi_{lk_l} \mathcal{N}\left(\mathbf{s}_t(l)|\phi_{lk_l}, \beta_{lk_l}\right) \right) \tag{2}$$

Split each data point into a missing part and an observed part: $\mathbf{x}_t^\top = (\mathbf{x}_t^{o\top}, \mathbf{x}_t^{m\top})$. In this paper, we only consider the random missing case [3], i.e. the probability for the missing entries $\mathbf{x}_t^m$ is independent of the value of $\mathbf{x}_t^m$, but could depend on the value of $\mathbf{x}_t^o$. The likelihood of the dataset is then defined to be

$$\mathcal{L}(\theta; \mathbf{X}) = \prod_t P(\mathbf{x}_t^o|\theta), \tag{3}$$

$$P(\mathbf{x}_t^o|\theta) = \int P(\mathbf{x}_t|\theta)\, d\mathbf{x}_t^m = \int \mathcal{N}(\mathbf{x}_t^o|[\mathbf{A}\mathbf{s}_t + \boldsymbol{\nu}]_t^o, [\boldsymbol{\Psi}]_t^o)P(\mathbf{s}_t|\theta_s)\, d\mathbf{s}_t \tag{4}$$

Here we have introduced the notation $[\cdot]_t^o$, which means taking only the observed dimensions (corresponding to the $t$th data point) of whatever is inside the square brackets. Since eqn. (4) is similar to eqn. (1), the variational Bayesian ICA [1, 2] can be extended naturally to handled missing data, but only if care is taken in discounting missing entries in the learning rules.

### 2.2 Variational Bayesian method

In a full Bayesian treatment, the posterior distribution of the parameters $\theta$ is obtained by

$$P(\theta|\mathbf{X}) = \frac{P(\mathbf{X}|\theta)P(\theta)}{P(\mathbf{X})} = \frac{\prod_t P(\mathbf{x}_t^o|\theta)P(\theta)}{P(\mathbf{X})} \tag{5}$$

where $P(\mathbf{X})$ is the marginal likelihood of the data and given as:

$$P(\mathbf{X}) = \int \prod_t P(\mathbf{x}_t^o|\theta)P(\theta)\, d\theta \tag{6}$$

The ICA model for $P(\mathbf{X})$ is defined with the following priors on the parameters $P(\theta)$,

$$P(A_{nl}) = \mathcal{N}(A_{nl}|0, \alpha_l) \qquad\qquad P(\boldsymbol{\pi}_l) = \mathcal{D}(\boldsymbol{\pi}_l|\mathbf{d}_o(\boldsymbol{\pi}_l))$$
$$P(\alpha_l) = \mathcal{G}(\alpha_l|a_o(\alpha_l), b_o(\alpha_l)) \qquad P(\phi_{lk_l}) = \mathcal{N}(\phi_{lk_l}|\mu_o(\phi_{lk_l}), \Lambda_o(\phi_{lk_l})) \tag{7}$$
$$P(\beta_{lk_l}) = \mathcal{G}(\beta_{lk_l}|a_o(\beta_{lk_l}), b_o(\beta_{lk_l}))$$
$$P(\nu_n) = \mathcal{N}(\nu_n|\mu_o(\nu_n), \Lambda_o(\nu_n)) \qquad P(\Psi_n) = \mathcal{G}(\Psi_n|a_o(\Psi_n), b_o(\Psi_n)) \tag{8}$$

where $\mathcal{N}(\cdot)$, $\mathcal{G}(\cdot)$ and $\mathcal{D}(\cdot)$ are the normal, gamma and Dirichlet distributions. $a_o(\cdot)$, $b_o(\cdot)$, $\mathbf{d}_o(\cdot)$, $\mu_o(\cdot)$, and $\Lambda_o(\cdot)$ are prechosen hyperparameters for the priors.

Under the variational Bayesian treatment, instead of performing the integration in eqn. (6) to solve for $P(\theta|\mathbf{X})$ directly, we approximate it by $Q(\theta)$ and opt to minimize the Kullback-Leibler distance between them:

$$-KL(Q(\theta)|P(\theta|\mathbf{X})) = \int Q(\theta) \log \frac{P(\theta|\mathbf{X})}{Q(\theta)} \, d\theta$$

$$= \int Q(\theta) \left[ \sum_t \log P(\mathbf{x}_t^o|\theta) + \log \frac{P(\theta)}{Q(\theta)} \right] d\theta - \log P(\mathbf{X}) \quad (9)$$

Since $-KL(Q(\theta)|P(\theta|\mathbf{X})) \leq 0$, we get a lower bound for the log marginal likelihood of the data,

$$\log P(\mathbf{X}) \geq \int Q(\theta) \sum_t \log P(\mathbf{x}_t^o|\theta) \, d\theta + \int Q(\theta) \log \frac{P(\theta)}{Q(\theta)} \, d\theta, \quad (10)$$

which can also be obtained by applying the Jensen's inequality to eqn. (6). $Q(\theta)$ is then solved by functional maximization of the lower bound. A separable approximate posterior $Q(\theta)$ will be assumed:

$$Q(\theta) = Q(\boldsymbol{\nu})Q(\boldsymbol{\Psi}) \times Q(\mathbf{A})Q(\boldsymbol{\alpha}) \times \prod_l \left[ Q(\boldsymbol{\pi}_l) \prod_{k_l} Q(\phi_{lk_l})Q(\beta_{lk_l}) \right]. \quad (11)$$

The second term in eqn. (10), which is the negative Kullback-Leibler divergence between approximate posterior $Q(\theta)$ and prior $P(\theta)$, can be expanded as,

$$\int Q(\theta) \log \frac{P(\theta)}{Q(\theta)} \, d\theta = \sum_l \int Q(\boldsymbol{\pi}_l) \log \frac{P(\boldsymbol{\pi}_l)}{Q(\boldsymbol{\pi}_l)} \, d\boldsymbol{\pi}_l$$

$$+ \sum_{l\,k_l} \int Q(\phi_{lk_l}) \log \frac{P(\phi_{lk_l})}{Q(\phi_{lk_l})} \, d\phi_{lk_l} + \sum_{l\,k_l} \int Q(\beta_{lk_l}) \log \frac{P(\beta_{lk_l})}{Q(\beta_{lk_l})} \, d\beta_{lk_l}$$

$$+ \iint Q(\mathbf{A})Q(\boldsymbol{\alpha}) \log \frac{P(\mathbf{A}|\boldsymbol{\alpha})}{Q(\mathbf{A})} \, d\mathbf{A} \, d\boldsymbol{\alpha} + \int Q(\boldsymbol{\alpha}) \log \frac{P(\boldsymbol{\alpha})}{Q(\boldsymbol{\alpha})} \, d\boldsymbol{\alpha}$$

$$+ \int Q(\boldsymbol{\nu}) \log \frac{P(\boldsymbol{\nu})}{Q(\boldsymbol{\nu})} \, d\boldsymbol{\nu} + \int Q(\boldsymbol{\Psi}) \log \frac{P(\boldsymbol{\Psi})}{Q(\boldsymbol{\Psi})} \, d\boldsymbol{\Psi} \quad (12)$$

### 2.3 Special treatment for missing data

Thus far the analysis follows almost exactly that of the variational Bayesian ICA on complete data, except that $P(\mathbf{x}_t|\theta)$ is replaced by $P(\mathbf{x}_t^o|\theta)$ in eqn. (6) and consequently the missing entries are discounted in the learning rules. However, it would be useful to obtain $Q(\mathbf{x}_t^m|\mathbf{x}_t^o)$, i.e., the approximate distribution on the missing entries, which is given by

$$Q(\mathbf{x}_t^m|\mathbf{x}_t^o) = \int Q(\theta) \int \mathcal{N}(\mathbf{x}_t^m|[\mathbf{A}\mathbf{s}_t + \boldsymbol{\nu}]_t^m, [\boldsymbol{\Psi}]_t^m)Q(\mathbf{s}_t) \, d\mathbf{s}_t \, d\theta. \quad (13)$$

As noted in [6], elements of $\mathbf{s}_t$ given $\mathbf{x}_t^o$ are dependent. More importantly, under the ICA model, $Q(\mathbf{s}_t)$ is unlikely to be a single Gaussian. This is evident from figure 1 which shows the probability density functions of the data $\mathbf{x}$ and hidden variable $\mathbf{s}$. The inserts show the sample data in the two spaces. Here the hidden sources assume density of $P(s_l) \propto \exp(-|s_l|^{0.7})$. They are mixed noiselessly to give $P(\mathbf{x})$ in the left graph. The cut in the left graph represents $P(x_1|x_2 = -0.5)$, which transforms into a highly correlated and non-Gaussian $P(\mathbf{s}|x_2 = -0.5)$.

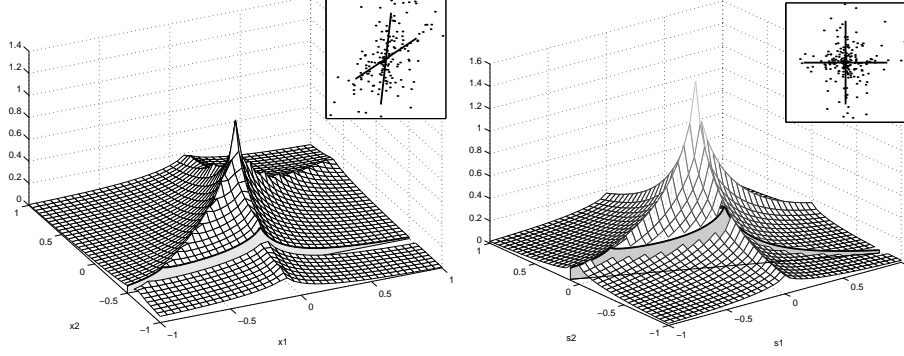

Figure 1: Pdfs for the data $\mathbf{x}$ (left) and hidden sources $\mathbf{s}$ (right). Inserts show the sample data in the two spaces. The "cuts" show $P(x_1|x_2 = -0.5)$ and $P(\mathbf{s}|x_2 = -0.5)$.

Unless we are interested only in the first and second order statistics of $Q(\mathbf{x}_t^m|\mathbf{x}_t^o)$, we should try to capture as much structure as possible of $P(\mathbf{s}_t|\mathbf{x}_t^o)$ in $Q(\mathbf{s}_t)$. In this paper, we take a slightly different route from [1, 2] when performing variational Bayesian learning. First, we break down $P(\mathbf{s}_t)$ (eqn. 2) into a mixture of $K^L$ Gaussians in the $L$ dimensional $\mathbf{s}$ space.

$$P(\mathbf{s}_t) = \sum_{k_1} \cdots \sum_{k_L} [\pi_{1k_1} \times \cdots \times \pi_{Lk_L} \times \mathcal{N}(\mathbf{s}_t(1)|\phi_{1k_1}\beta_{1k_1}) \times \cdots \times \mathcal{N}(\mathbf{s}_t(L)|\phi_{Lk_L}\beta_{Lk_L})]$$
$$= \sum_{\mathbf{k}} \pi_{\mathbf{k}} \mathcal{N}(\mathbf{s}_t|\phi_{\mathbf{k}}, \beta_{\mathbf{k}}) \tag{14}$$

Here we have defined $\mathbf{k}$ to be a vector index. The "$\mathbf{k}$th" Gaussian is centered at $\phi_{\mathbf{k}}$, of inverse covariance $\beta_{\mathbf{k}}$, in the source $\mathbf{s}$ space,

$$\pi_{\mathbf{k}} = \pi_{1k_1} \times \cdots \times \pi_{Lk_L} \qquad \phi_{\mathbf{k}} = (\phi_{1k_1}, \cdots, \phi_{lk_l}, \cdots, \phi_{Lk_L})^\top$$
$$\beta_{\mathbf{k}} = \operatorname{diag}(\beta_{1k_1}, \cdots \beta_{Lk_L}) \qquad \mathbf{k} = (k_1, \cdots, k_l, \cdots, k_L)^\top, \quad k_l = 1, \cdots, K \tag{15}$$

Log likelihood for $\mathbf{x}_t^o$ is then expanded using the Jensen's inequality,

$$\log P(\mathbf{x}_t^o|\theta) = \log \sum_{\mathbf{k}} \pi_{\mathbf{k}} \int P(\mathbf{x}_t^o|\mathbf{s}_t, \theta)\, \mathcal{N}(\mathbf{s}_t|\phi_{\mathbf{k}}, \beta_{\mathbf{k}})\, d\mathbf{s}_t$$
$$\geq \sum_{\mathbf{k}} Q(\mathbf{k}_t) \log \int P(\mathbf{x}_t^o|\mathbf{s}_t, \theta)\mathcal{N}(\mathbf{s}_t|\phi_{\mathbf{k}}, \beta_{\mathbf{k}})\, d\mathbf{s}_t + \sum_{\mathbf{k}} Q(\mathbf{k}_t) \log \frac{\pi_{\mathbf{k}}}{Q(\mathbf{k}_t)} \tag{16}$$

Here $Q(\mathbf{k}_t)$ is a short form for $Q(\mathbf{k}_t = \mathbf{k})$. $\mathbf{k}_t$ is a discrete hidden variable and $Q(\mathbf{k}_t = \mathbf{k})$ is the probability that the $t$th data point belongs to the $\mathbf{k}$th Gaussian. Recognizing that $\mathbf{s}_t$ is just a dummy variable, we introduce $Q(\mathbf{s}_{\mathbf{k}t})$, apply the Jensen's inequality again and get

$$\log P(\mathbf{x}_t^o|\theta) \geq \sum_{\mathbf{k}} Q(\mathbf{k}_t) \left[ \int Q(\mathbf{s}_{\mathbf{k}t}) \log P(\mathbf{x}_t^o|\mathbf{s}_{\mathbf{k}t}, \theta)\, d\mathbf{s}_{\mathbf{k}t} \right.$$
$$\left. + \int Q(\mathbf{s}_{\mathbf{k}t}) \log \frac{\mathcal{N}(\mathbf{s}_{\mathbf{k}t}|\phi_{\mathbf{k}}, \beta_{\mathbf{k}})}{Q(\mathbf{s}_{\mathbf{k}t})}\, d\mathbf{s}_{\mathbf{k}t} \right] + \sum_{\mathbf{k}} Q(\mathbf{k}_t) \log \frac{\pi_{\mathbf{k}}}{Q(\mathbf{k}_t)} \tag{17}$$

Substituting $\log P(\mathbf{x}_t^o|\theta)$ back into eqn. (10), the variational Bayesian method can be continued as usual. We have drawn in figure 2 a simplified graphical representation for the generative model of variational ICA. $\mathbf{x}_t$ is the observed variable, $\mathbf{k}_t$ and $\mathbf{s}_t$ are hidden variables and the rest are model parameters, where $\mathbf{k}_t$ indicates which of the $K^L$ expanded Gaussians generated $\mathbf{s}_t$.

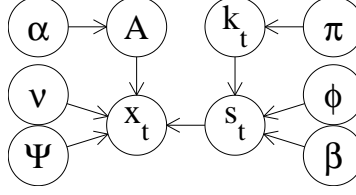

Figure 2: A simplified directed graph for the generative model of variational ICA. $\mathbf{x}_t$ is the observed variable, $\mathbf{k}_t$ and $\mathbf{s}_t$ are hidden variables and the rest are model parameters. The $\mathbf{k}_t$ indicates which of the $K^L$ expanded Gaussians generated $\mathbf{s}_t$.

## 3 Learning Rules

Combining eqns. (10,12 and 17) we perform functional maximization on the lower bound of the log marginal likelihood, $\log P(\mathbf{X})$, w.r.t. $Q(\theta)$ (eqn. 11), $Q(\mathbf{k}_t)$ and $Q(\mathbf{s}_{\mathbf{k}t})$ (eqn. 17) and obtain the following learning rules for the sufficient statistics of $Q(\theta)$ and $Q(\mathbf{s}_{\mathbf{k}t})$:

$$\Lambda(\nu_n) = \Lambda_o(\nu_n) + \langle\Psi_n\rangle \sum_t o_{nt}$$

$$\mu(\nu_n) = \frac{\Lambda_o(\nu_n)\mu_o(\nu_n) + \langle\Psi_n\rangle \sum_t o_{nt} \sum_{\mathbf{k}} Q(\mathbf{k}_t)\langle(x_{nt} - \mathbf{A}_{n\cdot}\mathbf{s}_{\mathbf{k}t})\rangle}{\Lambda(\nu_n)} \tag{18}$$

$$a(\Psi_n) = a_o(\Psi_n) + \frac{1}{2}\sum_t o_{nt}$$

$$b(\Psi_n) = b_o(\Psi_n) + \frac{1}{2}\sum_t o_{nt} \sum_{\mathbf{k}} Q(\mathbf{k}_t)\langle(x_{nt} - \mathbf{A}_{n\cdot}\mathbf{s}_{\mathbf{k}t} - \nu_n)^2\rangle \tag{19}$$

$$\mathbf{\Lambda}(\mathbf{A}_{n\cdot}) = \text{diag}\left(\langle\alpha_1\rangle, \cdots \langle\alpha_L\rangle\right) + \langle\Psi_n\rangle \sum_t o_{nt} \sum_{\mathbf{k}} Q(\mathbf{k}_t)\langle\mathbf{s}_{\mathbf{k}t}\mathbf{s}_{\mathbf{k}t}^\top\rangle$$

$$\boldsymbol{\mu}(\mathbf{A}_{n\cdot}) = \left(\langle\Psi_n\rangle \sum_t o_{nt}(x_{nt} - \langle\nu_n\rangle) \sum_{\mathbf{k}} Q(\mathbf{k}_t)\langle\mathbf{s}_{\mathbf{k}t}^\top\rangle\right) \mathbf{\Lambda}(\mathbf{A}_{n\cdot})^{-1} \tag{20}$$

$$a(\alpha_l) = a_o(\alpha_l) + \frac{N}{2} \qquad b(\alpha_l) = b_o(\alpha_l) + \frac{1}{2}\sum_n \langle A_{nl}^2\rangle \tag{21}$$

$$d(\pi_{lk}) = d_o(\pi_{lk}) + \sum_t \sum_{\mathbf{k}_l=k} Q(\mathbf{k}_t) \tag{22}$$

$$\Lambda(\phi_{lk_l}) = \Lambda_o(\phi_{lk_l}) + \langle\beta_{lk_l}\rangle \sum_t \sum_{\mathbf{k}_l=k} Q(\mathbf{k}_t)$$

$$\mu(\phi_{lk_l}) = \frac{\Lambda_o(\phi_{lk_l})\mu_o(\phi_{lk_l}) + \langle\beta_{lk_l}\rangle \sum_t \sum_{\mathbf{k}_l=k} Q(\mathbf{k}_t)\langle s_{\mathbf{k}t}(l)\rangle}{\Lambda(\phi_{lk_l})} \tag{23}$$

$$a(\beta_{lk_l}) = a_o(\beta_{lk_l}) + \frac{1}{2}\sum_t \sum_{\mathbf{k}_l=k} Q(\mathbf{k}_t)$$

$$b(\beta_{lk_l}) = b_o(\beta_{lk_l}) + \frac{1}{2}\sum_t \sum_{\mathbf{k}_l=k} Q(\mathbf{k}_t)\langle(s_{\mathbf{k}t}(l) - \phi_{lk_l})^2\rangle \tag{24}$$

$$Q(\mathbf{s}_{\mathbf{k}t}) = \mathcal{N}(\mathbf{s}_{\mathbf{k}t}|\boldsymbol{\mu}(\mathbf{s}_{\mathbf{k}t}), \mathbf{\Lambda}(\mathbf{s}_{\mathbf{k}t}))$$

$$\mathbf{\Lambda}(\mathbf{s}_{\mathbf{k}t}) = \text{diag}\left(\langle\beta_{1\mathbf{k}_1}\rangle, \cdots \langle\beta_{L\mathbf{k}_L}\rangle\right) + \langle\mathbf{A}^\top \text{diag}\left(o_{1t}\Psi_1, \cdots o_{Nt}\Psi_N\right) \mathbf{A}\rangle \tag{25}$$

$$\mathbf{\Lambda}(\mathbf{s}_{\mathbf{k}t})\boldsymbol{\mu}(\mathbf{s}_{\mathbf{k}t}) = \langle\beta_{1\mathbf{k}_1}\phi_{1\mathbf{k}_1}, \cdots \beta_{L\mathbf{k}_L}\phi_{L\mathbf{k}_L}\rangle^\top + \langle\mathbf{A}^\top \text{diag}\left(o_{1t}\Psi_1, \cdots o_{Nt}\Psi_N\right)(\mathbf{x}_t - \boldsymbol{\nu})\rangle$$

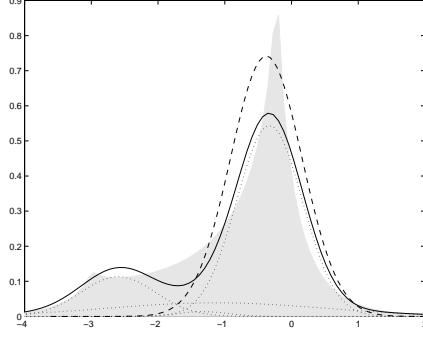

Figure 3: The approximation of $Q(\mathbf{x}_t^m|\mathbf{x}_t^o)$ from the full missing ICA (solid line) and the polynomial missing ICA (dashed line). Shaded area is the exact posterior $P(\mathbf{x}_t^m|\mathbf{x}_t^o)$ corresponding to the noiseless mixture in fig. 1 with observed $x_2{=}{-}2$. Dotted lines are contribution from the individual $Q(\mathbf{x}_{\mathbf{k}t}^m|\mathbf{x}_t^o, \mathbf{k})$.

In the above equations, $\langle \cdot \rangle$ denotes the expectation over the posterior distributions $Q(\cdot)$. $\mathbf{A}_{n\cdot}$ is the $n$th row of the mixing matrix $\mathbf{A}$, $\sum_{\mathbf{k}_l=k}$ means picking out those Gaussians such that the $l$th element of their indices $\mathbf{k}$ has the value of $k$, and $o_{nt}$ is a binary indicator variable for whether or not $x_{nt}$ is observed.

For a model of equal noise variance among all the observation dimensions, the summation in the learning rules for $Q(\mathbf{\Psi})$ would be over both $t$ and $n$. Note that there exists scale and translational degeneracy in the model, as given by eqn. (1) and (2). After each update of $Q(\boldsymbol{\pi}_l)$, $Q(\phi_{lk_l})$ and $Q(\beta_{lk_l})$, it is better to rescale $P(\mathbf{s}_t(l))$ to have zero mean and unit variance. $Q(\mathbf{s}_{\mathbf{k}t})$, $Q(\mathbf{A})$, $Q(\boldsymbol{\alpha})$, $Q(\boldsymbol{\nu})$ and $Q(\mathbf{\Psi})$ have to be adjusted correspondingly. Finally, $Q(\mathbf{k}_t)$ is given by,

$$\log Q(\mathbf{k}_t) = \langle \log P(\mathbf{x}_t^o|\mathbf{s}_{\mathbf{k}t}, \theta) + \log \mathcal{N}(\mathbf{s}_{\mathbf{k}t}|\boldsymbol{\phi}_{\mathbf{k}}, \boldsymbol{\beta}_{\mathbf{k}}) - \log Q(\mathbf{s}_{\mathbf{k}t}) + \log \boldsymbol{\pi}_{\mathbf{k}} \rangle - \log z_t \quad (26)$$

where $z_t$ is a normalization constant. The lower bound $\mathcal{E}(\mathbf{X}, Q(\theta)|\mathcal{H})$ for the log marginal likelihood

$$\mathcal{E}(\mathbf{X}, Q(\theta)|\mathcal{H}) = \sum_t \log z_t + \int Q(\theta) \log \frac{P(\theta)}{Q(\theta)} \, d\theta \quad (27)$$

can be monitored during learning and used for comparison of different solutions or models.

## 4   Filling in missing entries

The approximate distribution $Q(\mathbf{x}_t^m|\mathbf{x}_t^o)$ can be obtained by a summation of $Q(\mathbf{x}_{\mathbf{k}t}^m|\mathbf{x}_t^o, \mathbf{k})$:

$$Q(\mathbf{x}_t^m|\mathbf{x}_t^o) = \sum_{\mathbf{k}} Q(\mathbf{k}_t) \int \delta(\mathbf{x}_t^m - \mathbf{x}_{\mathbf{k}t}^m) Q(\mathbf{x}_{\mathbf{k}t}^m|\mathbf{x}_t^o, \mathbf{k}) \, d\mathbf{x}_{\mathbf{k}t}^m \;, \quad (28)$$

$$Q(\mathbf{x}_{\mathbf{k}t}^m|\mathbf{x}_t^o, \mathbf{k}) = \int Q(\theta) \int \mathcal{N}(\mathbf{x}_{\mathbf{k}t}^m|[\mathbf{A}\mathbf{s}_{\mathbf{k}t} + \boldsymbol{\nu}]_t^m, [\mathbf{\Psi}]_t^m) Q(\mathbf{s}_{\mathbf{k}t}) \, d\mathbf{s}_{\mathbf{k}t} \, d\theta \quad (29)$$

Estimation of $Q(\mathbf{x}_t^m|\mathbf{x}_t^o)$ using the above equations is demonstrated in fig. 3. The shaded area is the exact posterior $P(\mathbf{x}_t^m|\mathbf{x}_t^o)$ for the noiseless mixing in fig. 1 with observed $x_2{=}{-}2$ and the solid line is the approximation by eqn. 28–29. We have modified the variational ICA of [1] by discounting missing entries in the learning rules. The dashed line is the approximation of $Q(\mathbf{x}_t^m|\mathbf{x}_t^o)$ from this modified method. The treatment of fully expanding the $K^L$ hidden source Gaussians discussed in section 2.3 is called "full missing ICA", and the modified method is "polynomial missing ICA". The "full missing ICA" gives a more accurate fit for $P(\mathbf{x}_t^m|\mathbf{x}_t^o)$ and a better estimate for $\langle \mathbf{x}_t^m|\mathbf{x}_t^o \rangle$.

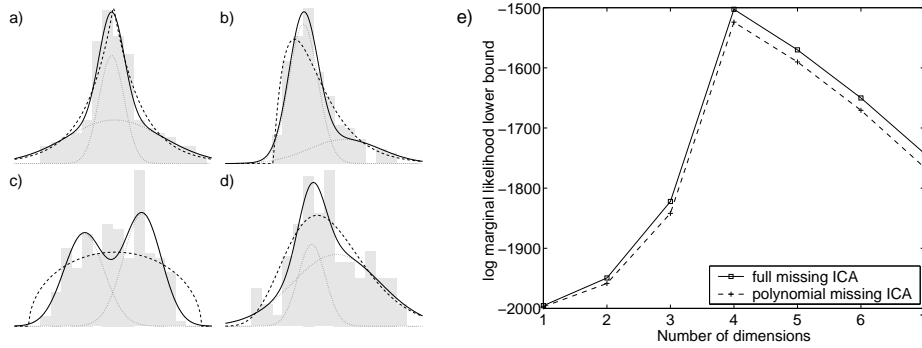

Figure 4: a)-d) Source density modeling by variational missing ICA of the synthetic data. Histograms: recovered sources distribution; dashed lines: original probability densities; solid line: mixture of Gaussians modeled probability densities; dotted lines: individual Gaussian contribution. e) $\mathcal{E}(\mathbf{X}, Q(\theta)|\mathcal{H})$ as a function of hidden source dimensions.

## 5 Experiment

### 5.1 Synthetic Data

In the first experiment, 200 data points were generated by mixing 4 sources randomly in a 7 dimensional space. The generalized Gaussian, gamma and beta distributions were used to represent source densities of various skewness and kurtosis (fig. 4 a)-d)). Noise at –26 dB level was added to the data and missing entries were created with a probability of 0.3. In fig. 4 a)-d), we plotted the histograms of the recovered sources and the probability density functions (pdf) of the 4 sources. The dashed line is the exact pdf used to generate the data and solid line is the modeled pdf by mixture of two 1-D Gaussians (eqn. 2). Fig. 4 e) plots the lower bound of log marginal likelihood (eqn. 27) for models assuming different numbers of intrinsic dimensions. As expected, the Bayesian treatment allows us to the infer the intrinsic dimension of the data cloud. In the figure, we also plot the $\mathcal{E}(\mathbf{X}, Q(\theta)|\mathcal{H})$ from the polynomial missing ICA. It is clear that the full missing ICA gave a better fit to the data density. Furthermore, the polynomial missing ICA converges slower per epoch of learning, suffers from many more local minima and problems get worse with higher missing rate.

### 5.2 Mixing Images

This experiment demonstrates the ability of the proposed method to fill in missing values while performing demixing. The 1st column in fig. 5 shows the 2 original 380-by-380 pixels images. They were linearly mixed into 3 images and –20 dB noise was added. 20% missing entries were introduced randomly. The denoised mixtures and recovered sources are in the 3rd and 4th columns of fig. 5. 0.8% of the pixels were missing from all 3 mixed images and could not be recovered. 38.4% of the pixels were missing from only 1 mixed image and could be filled in with low uncertainty. 9.6% of the pixels were missing from any two of the mixed images. Estimation of their values incurred high uncertainty. From fig. 5, we can see that the source images were well separated and the mixed images were nicely denoised. The denoised mixed images in this example were only meant to visually illustrate the method. However, if $(x_1, x_2, x_3)$ represent cholesterol, blood sugar and uric acid level, for example, it would be possible to fill in the third when only two are available.

## 6 Conclusion

In this paper, we derived the learning rules for variational Bayesian ICA with missing data. The complexity of the method is exponential in $L$. However, this exponential growth in

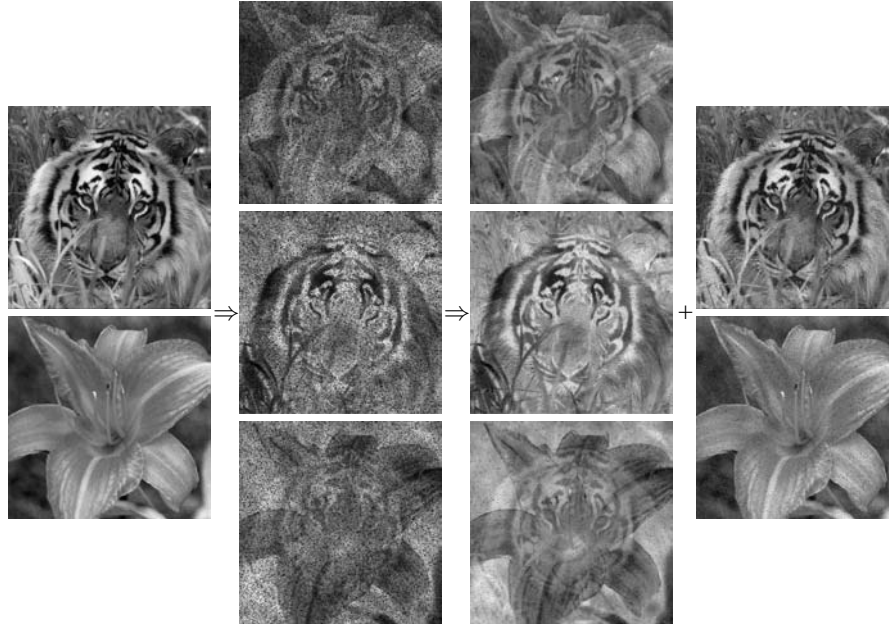

Figure 5: A demonstration of recovering missing values. The original images are in the 1st column. 20% of the pixels in the mixed images (2nd column) are missing, while only 0.8% are missing from the denoised mixed (3rd column) and separated images (4th column).

complexity is manageable and worthwhile for small data sets containing missing entries in a high dimensional space. The proposed method shows promise in analyzing and identifying projections of datasets that have a very limited number of expensive data points yet contain missing entries due to data scarcity. We have applied the variational missing ICA to a primates brain volumetric dataset containing 44 examples in 57 dimensions. Very encouraging results were obtained and will be reported in another paper.

## References

[1] Kwokleung Chan, Te-Won Lee, and Terrence J. Sejnowski. Variational learning of clusters of undercomplete nonsymmetric independent components. *Journal of Machine Learning Research*, 3:99–114, 2002.

[2] Rizwan A. Choudrey and Stephen J. Roberts. Flexible Bayesian independent component analysis for blind source separation. In *3rd International Conference on Independent Component Analysis and Blind Signal Separation*, pages 90–95, San Diego, Dec. 09-12 2001.

[3] Z. Ghahramani and M. Jordan. Learning from incomplete data. Technical Report CBCL Paper No. 108, Center for Biological and Computational Learning, Massachusetts Institute of Technology, 1994.

[4] Aapo Hyvarinen, Juha Karhunen, and Erkki Oja. *Independent Component Analysis*. J. Wiley, New York, 2001.

[5] R. J. A. Little and D. B. Rubin. *Statistical Analysis with Missing Data*. Wiley, New York, 1987.

[6] Max Welling and Markus Weber. Independent component analysis of incomplete data. In *1999 6th Joint Symposium on Neural Compuatation Proceedings*, volume 9, pages 162–168. UCSD, May. 22 1999.
